# A Phase Space Approach to Minimax Entropy Learning and the Minutemax Approximations

**James M. Coughlan**
Smith-Kettlewell Inst.
San Francisco, CA 94115

**A. L. Yuille**
Smith-Kettlewell Inst.
San Francisco, CA 94115

## Abstract

There has been much recent work on measuring image statistics and on learning probability distributions on images. We observe that the mapping from images to statistics is many-to-one and show it can be quantified by a phase space factor. This phase space approach throws light on the Minimax Entropy technique for learning Gibbs distributions on images with potentials derived from image statistics and elucidates the ambiguities that are inherent to determining the potentials. In addition, it shows that if the phase factor can be approximated by an analytic distribution then this approximation yields a swift "Minutemax" algorithm that vastly reduces the computation time for Minimax entropy learning. An illustration of this concept, using a Gaussian to approximate the phase factor, gives a good approximation to the results of Zhu and Mumford (1997) in just seconds of CPU time. The phase space approach also gives insight into the multi-scale potentials found by Zhu and Mumford (1997) and suggests that the forms of the potentials are influenced greatly by phase space considerations. Finally, we prove that probability distributions learned in feature space alone are equivalent to Minimax Entropy learning with a multinomial approximation of the phase factor.

## 1  Introduction

Bayesian probability theory gives a powerful framework for visual perception (Knill and Richards 1996). This approach, however, requires specifying prior probabilities and likelihood functions. Learning these probabilities is difficult because it requires estimating distributions on random variables of very high dimensions (for example, images with $200 \times 200$ pixels, or shape curves of length 400 pixels). An important

recent advance is the Minimax Entropy Learning theory. This theory was developed by Zhu, Wu and Mumford (1997 and 1998) and enables them to learn probability distributions for the intensity properties and shapes of natural stimuli and clutter. In addition, when applied to real world images it has an interesting link to the work on natural image statistics (Field 1987), (Ruderman and Bialek 1994), (Olshaussen and Field 1996). We wish to simplify Minimax and make the learning easier, faster and more transparent.

In this paper we present a phase space approach to Minimax Entropy learning. This approach is based on the observation that the mapping from images to statistics is many-to-one and can be quantified by a phase space factor. If this phase space factor can be approximated by an analytic function then we obtain approximate "Minutemax" algorithms which greatly speed up the learning process. In one version of this approximation, the unknown parameters of the distribution to be learned are related linearly to the empirical statistics of the image data set, and may be solved for in seconds or less. Independent of this approximation, the Minutemax framework also illuminates an important combinatoric aspect of Minimax, namely the fact that many different images can give rise to the same image statistics. This "phase space" factor explains the ambiguities inherent in learning the parameters of the unknown distribution, and motivates the approximation that reduces the problem to linear algebra. Finally, we prove that probability distributions learned in feature space alone are equivalent to Minimax Entropy learning with a multinomial approximation of the phase factor.

## 2   A Phase Space Perspective on Minimax

We wish to learn a distribution $P(\mathbf{I})$ on images, where $\mathbf{I}$ denotes the set of pixel values $I(x, y)$ on a finite image lattice, and each value $I(x, y)$ is quantized to a finite set of intensity values. (In fact, this approach is general and applies to any patterns, not just images.) We define a set of image statistics $\phi_1(\mathbf{I}), \phi_2(\mathbf{I}), \ldots, \phi_S(\mathbf{I})$, which we concatenate as a single vector function $\vec{\phi}(\mathbf{I})$. If these statistics have empirical mean $\vec{d} \equiv <\vec{\phi}(\mathbf{I})>$ on a dataset of images (we assume a large enough dataset for the law of large numbers to apply; see Zhu and Mumford (1997) for an analysis of the errors inherent in this assumption) then the maximum entropy distribution $P_M(\mathbf{I})$ with these empirical statistics is an exponential (Gibbs) distribution of the form

$$P_M(\mathbf{I}) = \frac{e^{\vec{\lambda} \cdot \vec{\phi}(\mathbf{I})}}{Z(\vec{\lambda})}, \tag{1}$$

where the potential $\vec{\lambda}$ is set so that $<\vec{\phi}(\mathbf{I})>_M = \vec{d}$.

In summary, the goal of Minimax Learning is to to find an appropriate set of image filters for the domain of interest (i.e. maximally informative filters) and to estimate $\vec{\lambda}$ given $\vec{d}$. Extensive computation is required to determine $\vec{\lambda}$; the phase space approach to Minimax Learning motivates approximations that make $\vec{\lambda}$ easy to estimate.

### 2.1   Image Histogram Statistics

The statistics we consider (following Zhu, Wu and Mumford (1997, 1998)) are defined as histograms of the responses of one or more filters applied across an entire image. Consider a single filter $f$ (linear or non-linear) with response $f_{\mathbf{x}}(\mathbf{I})$ centered at position $\mathbf{x}$ in the image. Without loss of generality, we will assume the filter has quantized integer responses from 1 through $f_{max}$.

For notational convenience we transform the filter response $f_{\mathbf{x}}(\mathbf{I})$ to a binary representation $\vec{b}_{\mathbf{x}}(\mathbf{I})$, defined as a column vector with $f_{max}$ components: $\vec{b}_{\mathbf{x},z}(\mathbf{I}) = \delta_{z,f_{\mathbf{x}}(\mathbf{I})}$, where index $z$ ranges from 1 through $f_{max}$. This vector is composed of all zeros except for the entry corresponding to the filter response, which is set to one. The image statistics vector is then a histogram vector defined as the average of the $\vec{b}_{\mathbf{x}}(\mathbf{I})$'s over all $N$ pixels: $\vec{\phi}(\mathbf{I}) = \frac{1}{N}\sum_{\mathbf{x}} \vec{b}_{\mathbf{x}}(\mathbf{I})$. The entries in $\vec{\phi}(\mathbf{I})$ then sum to 1. (We can generalize to the case of multiple filters $f^{(1)}, f^{(2)}, \ldots, f^{(m)}$, as detailed in Coughlan and Yuille (1999).)

## 2.2 The Phase Factor

The original Minimax distribution $P_M(\mathbf{I})$ induces a distribution $P_M(\vec{\phi})$ on the statistics themselves, without reference to a particular image:

$$P_M(\vec{\phi}_0) = \sum_{\mathbf{I}} \delta_{\vec{\phi}_0,\vec{\phi}(\mathbf{I})} P_M(\mathbf{I}) = g(\vec{\phi}_0)\frac{e^{\vec{\lambda}\cdot\vec{\phi}_0}}{Z(\vec{\lambda})} \tag{2}$$

where $g(\vec{\phi})$ is a combinatoric *phase space factor*, with a corresponding *normalized* combinatoric distribution $\hat{g}(\vec{\phi})$, defined by:

$$g(\vec{\phi}_0) = \sum_{\mathbf{I}} \delta_{\vec{\phi}_0,\vec{\phi}(\mathbf{I})}, and \ \hat{g}(\vec{\phi}) = g(\vec{\phi})/Q^N, \tag{3}$$

where the phase space factor $g(\vec{\phi})$ counts the number of images $\mathbf{I}$ having statistics $\vec{\phi}$. $N$ is the number of pixels and $Q$ is the number of pixel intensity levels, i.e. $Q^N$ is the total number of possible images $\mathbf{I}$. It should be emphasized that *the phase factor depends only on the set of filters chosen and is independent of the true distribution $P(\mathbf{I})$*. Thus the phase factor can be computed offline, independent of the image data set.

In this paper we will discuss two useful approximations to $g(\vec{\phi})$: a Gaussian approximation, which yields the swift approximation for learning, and a multinomial approximation, which establishes a connection between Minimax and standard feature learning.

## 2.3 The Non-Uniqueness of the Potential $\vec{\lambda}$

Given a set of filters and their empirical mean statistics $\vec{d}$, is the potential $\vec{\lambda}$ uniquely specified? Clearly, any solution for $\vec{\lambda}$ may be shifted by an additive constant ($\lambda_i \to \lambda_i' = \lambda_i + k$ for all $i$), yielding a different normalization constant $Z(\vec{\lambda'})$ but preserving $P_M(\mathbf{I})$. In this section we show that other, non-trivial ambiguities in $\vec{\lambda}$ which preserve $P_M(\mathbf{I})$ can exist, stemming from the fact that some values of $\vec{\phi}$ are inconsistent with every possible image $\mathbf{I}$ and hence never arise (in *any* possible image dataset). These "intrinsic" ambiguities are inherent to Minimax and are independent of the true distribution $P(\mathbf{I})$. We will also discuss a second type of possible ambiguity which depends on the characteristics of the image dataset used for learning.

We can uncover the intrinsic ambiguities in $\vec{\lambda}$ by examining the covariance $C$ of $\hat{g}(\vec{\phi})$. (See Coughlan and Yuille (1999) for details on calculating the mean $\vec{c}$ and covariance $C$ for any set of linear filters or non-linear filters that are scalar functions

of linear filters.) Defining the set of all possible statistics values $\Phi = \{\vec{\phi} : g(\vec{\phi}) \neq 0\}$, the null space of $C$ reflects degeneracy (i.e. flatness) in $\Phi$. The following theorem, proved in Coughlan and Yuille (1999), shows that $\vec{\lambda}$ is determined only up to a hyperplane whose dimension is the nullity of $C$.

**Theorem 1 (Intrinsic Ambiguity in $\vec{\lambda}$).** $C\vec{\mu} = 0$ if and only if $e^{(\vec{\lambda}+\vec{\mu})\cdot\vec{\phi}(\mathbf{I})}/Z(\vec{\lambda}+\vec{\mu})$ and $e^{\vec{\lambda}\cdot\vec{\phi}(\mathbf{I})}/Z(\vec{\lambda})$ are identical distributions on $\mathbf{I}$.

In addition to this intrinsic ambiguity in $\vec{\lambda}$, it is also possible that different values of $\vec{\lambda}$ may yield distinct distributions which nevertheless have the same mean statistics $< \vec{\phi} >$ on the image dataset. (As shown in Coughlan and Yuille (1999), there is a *convex set of distributions*, of which the true distribution $P(\mathbf{I})$ is a member, which share the same mean statistics $< \vec{\phi} >$.) This second kind of ambiguity stems from the fact that the mean statistics convey only a fraction of the information that is contained in the true distribution $P(\mathbf{I})$. To resolve this second ambiguity it is necessary to extract more information from the image data set. The simplest way to achieve this is to use a larger (or more informative) set of filters to lower the entropy of $P_M(\mathbf{I})$ (this topic is discussed in more detail in Zhu, Wu and Mumford (1997, 1998), Coughlan and Yuille (1999)). Alternatively, one can extend Minimax to include second-order statistics, i.e. the covariance of $\vec{\phi}$ in addition to its mean $\vec{d}$. This is an important topic for future research.

## 3   The Minutemax Approximations

We now illustrate the phase space approach by showing that suitable approximations of the phase space factor $g(\vec{\phi})$ make it easy to estimate the potential $\vec{\lambda}$ given the empirical mean $\vec{d}$. The resulting fast approximations to Minimax Learning are called "Minutemax" algorithms.

### 3.1   The Gaussian Approximation of $g(\vec{\phi})$

If the phase space factor $g(\vec{\phi})$ may be approximated as a multi-variate Gaussian (see Coughlan and Yuille (1999) for a justification of this approximation) then the probability distribution $P_M(\vec{\phi}) = g(\vec{\phi})e^{\vec{\lambda}\cdot\vec{\phi}}/Z(\vec{\lambda})$ reduces to another multi-variate Gaussian. (Note that we are making the Gaussian approximation in $\vec{\phi}$ space–the space of all possible image statistics histograms–and not filter response (feature) space.) As we will see, this result greatly simplifies the problem of estimating the potential $\vec{\lambda}$.

Recall that the mean and covariance of $\hat{g}(\vec{\phi})$ are denoted by $\vec{c}$ and $C$, respectively. The null space of $C$ has dimension $n$ and is spanned by vectors $\vec{u}^{(1)}, \vec{u}^{(2)} \ldots \vec{u}^{(n)}$. As discussed in Theorem 1, for all *feasible* values of $\vec{\phi}$ (i.e. all $\vec{\phi} \in \Phi$) and all $\vec{\mu}$ in the null space, $\vec{\mu} \cdot \vec{\phi}$ is a constant $k$. Thus we have that

$$g_{gauss}(\vec{\phi}) \propto \{\prod_{i=1}^{n} \delta_{\vec{\phi}\cdot\vec{u}_i,k}\}e^{-\frac{1}{2}(\vec{\phi}_r-\vec{c}_r)^T C_r^{-1}(\vec{\phi}_r-\vec{c}_r)}, \qquad (4)$$

where the subscript $r$ denotes projection onto the rank of $C$. Thus $P_{gauss}(\vec{\phi}) \propto g_{gauss}(\vec{\phi})e^{\vec{\lambda}\cdot\vec{\phi}} \propto \{\prod_{i=1}^{n} \delta_{\vec{\phi}\cdot\vec{u}_i,k}\}e^{-\frac{1}{2}(\vec{\phi}_r-\vec{c}_r)^T C_r^{-1}(\vec{\phi}_r-\vec{c}_r)+\vec{\lambda}\cdot\vec{\phi}}$. Completing the square in the exponent yields $P_{gauss}(\vec{\phi}) \propto \{\prod_{i=1}^{n} \delta_{\vec{\phi}\cdot\vec{u}_i,k}\}e^{-\frac{1}{2}(\vec{\phi}_r-\vec{\psi}_r)^T C_r^{-1}(\vec{\phi}_r-\vec{\psi}_r)}$ where $\vec{\psi}_r$

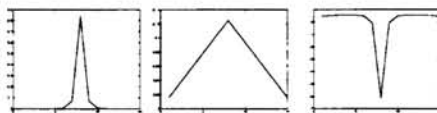

Figure 1: From left to right: $\vec{d}$, $\vec{c}$ and $-\vec{\lambda}$ (as computed by the Gaussian Minutemax approximation) for first filter alone.

is the projection of any $\vec{\psi}$ that satisfies $\vec{\psi} = \vec{c} + C\vec{\lambda}$. Since $P_{gauss}(\vec{\phi})$ is a Gaussian we have $<\vec{\phi}>_{gauss} = \vec{\psi} = \vec{d}$, and so we can write a linear equation relating $\vec{\lambda}$ and $\vec{d}$: $\vec{d} = \vec{c} + C\vec{\lambda}$.

It can be shown (Zhu – private communication) that solving this equation is equivalent to one step of Newton-Raphson for minimization of an appropriate cost function. This will fail to be a good approximation if the cost function is highly non-quadratic. As explained in Coughlan and Yuille (1999), the Gaussian approximation is also equivalent to a second-order perturbation expansion of the partition function $Z(\vec{\lambda})$; higher-order corrections can be made by computing higher-order moments of $g(\vec{\phi})$.

### 3.2  Experimental Results

We tested the Gaussian Minutemax procedure on two sets of filters: a single (fine scale) image gradient filter $\partial I/\partial x$, and a set of multi-scale image gradient filters defined at three scales, similar to those used by Zhu and Mumford (1997). In both sets, the fine scale gradient filter is linear with kernel $(1, -1)$, representing a discretization of $\partial/\partial x$. In the second set, the medium scale filter kernel is $(U_2, -U_2)/4$ and the coarse scale kernel is $(U_4, -U_4)/16$, where $U_n$ denotes the $n \times n$ matrix of all ones. The responses of the medium and coarse filters were rounded (i.e. quantized) to the nearest integer, thus adding a non-linearity to these filters. Finally, $\vec{d}$ was measured on a data set of over 100 natural images; the fine scale components of $\vec{d}$ are shown in the first panel of Figure (1) and were empirically very similar to the medium and coarse scale components.

A $\vec{\lambda}$ that solves $\vec{d} = \vec{c} + C\vec{\lambda}$ is shown in the third panel of Figure (1) for the first filter (along with $\vec{c}$ in the second panel) and in the three panels of Figure (2) for the multi-scale filter set. The form of $\vec{\lambda}$ is qualitatively similar to that obtained by Zhu and Mumford (1997) (bearing in mind that Zhu disregarded any filter responses with magnitude above $Q/2$, i.e. his filter response range is half of ours). In addition, the eigenvectors of $C$ with small eigenvalues are large away from the origin, so one should not trust the values of the potentials there (obtained by *any* algorithm).

Zhu and Mumford (1997) report interactions between filters`applied at different scales. This is because the resulting potentials appear different than the potential at the fine scale even though the histograms appear similar at all scales. We argue, however, that some of this "interaction" is due to *the different phase factors at different scales*. In other words the potentials would look different at different scales *even if the empirical histograms were identical* because of differing phase factors.

### 3.3  The Multinomial Approximation of $g(\vec{\phi})$

Many learning theories simply make probability distributions on feature space. How do they differ from Minimax Entropy Learning which works on image space? By

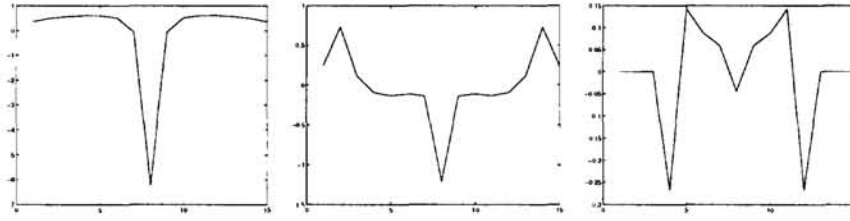

Figure 2: From left to right: the fine, medium and coarse components of $-\vec{\lambda}$ as computed by the Gaussian Minutemax approximation.

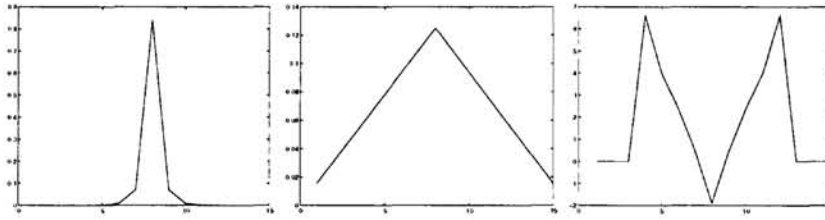

Figure 3: Left to right: $\vec{d}$, $\vec{c}$, and $-\vec{\lambda}$ as given by multinomial approximation for the $\partial/\partial x$ filter at fine scale.

examining the phase factor we will show that the two approaches are not identical in general. The feature space learning ignores the coupling between the filters which arise due to how the statistics are obtained. More precisely, the probability distribution obtained on feature space, $P_F$, is equivalent to the Minimax distribution $P_M$ *if, and only if, the phase factor is multinomial*.

We begin the analysis by considering a single filter. As before we define the combinatoric mean $\vec{c} = \sum_{\vec{\phi}} \hat{g}(\vec{\phi})\vec{\phi}$. The multinomial approximation of $\hat{g}(\vec{\phi})$ is equivalent to assuming that the combinatoric frequencies of filter responses are *independent* from pixel to pixel. Since the combinatoric frequency of filter response $j \in \{1, 2, \ldots, f_{max}\}$ is $c_j$ and there are $N\phi_j$ pixels with response $j$, we have:

$$\hat{g}_{mult}(\vec{\phi}) = \prod_{j=1}^{f_{max}} c_j^{N\phi_j} \frac{N!}{\prod_{j=1}^{f_{max}} (N\phi_j)!}, \;\; and \;\; P_{mult}(\vec{\phi}) \propto \prod_{j=1}^{f_{max}} (c_j e^{\lambda_j/N})^{N\phi_j} \frac{N!}{\prod_{j=1}^{f_{max}} (N\phi_j)!}, \tag{5}$$

using $P_{mult}(\vec{\phi}) \propto g_{mult}(\vec{\phi})e^{\vec{\lambda}\cdot\vec{\phi}}$. Therefore $P_{mult}(\vec{\phi})$ is also a multinomial. Shifting the $\lambda_j$'s by an appropriate additive constant, we can make the constant of proportionality in the above equation equal to 1. In this case we have $< \phi_j >_{mult} = c_j e^{\lambda_j/N}$ and $\lambda_j = N\log(d_j/c_j)$ by setting $< \phi_j >_{mult}$ to the empirical mean $d_j$.

Note that if any component $d_j$ of the empirical mean is close to 0 then by the previous equation any small perturbations in $d_j$ (e.g. from sampling error) will yield large changes in $\lambda_j$, making the estimate of that component unstable.

We can generalize the multinomial approximation of $g(\vec{\phi})$ to the multiple filter case merely by factoring $\hat{g}_{mult}(\vec{\phi})$ into separate multinomials, one for each filter. Of course, this approximation neglects all interactions among filters (and among pixels).

### 3.4 The Multinomial Approximation and Feature Learning

The connection between the multinomial approximation and feature learning is straightforward once we consider a distribution on the feature vector $\vec{f}$. This distribution (denoted $P_F$ for "feature") is constructed assuming independent filter responses from pixel to pixel and with statistics matching the empirical mean $\vec{d}$: $P_F(\vec{f}) = \prod_{i=1}^{N} d_{(f_i)}$, where $f_i$ denotes the filter response at pixel $i$. Then it follows that $P_F(\vec{\phi})$ is a multinomial: $P_F(\vec{\phi}) = \prod_{j=1}^{f_{max}} d_j^{N\phi_j} \frac{N!}{\prod_{j=1}^{f_{max}}(N\phi_j)!}$. Since $d_j = c_j e^{\lambda_j/N}$, we have our main result that $P_F(\vec{\phi}) = P_{mult}(\vec{\phi})$.

## 4 Conclusion

The main point of this paper is to introduce the phase space factor to quantify the mapping between images and their feature statistics. This phase space approach can: (i) provide fast approximate "Minutemax" algorithms, (ii) clarify the relationship between probability distributions learned in feature and image space, and (iii) to determine intrinsic ambiguities in the $\vec{\lambda}$ potentials.

### Acknowledgements

We acknowledge stimulating discussions with Song Chun Zhu. Funding was provided by the Smith-Kettlewell Institute Core Grant and the Center for Imaging Sciences ARO grant DAANO4-95-1-0494.

### References

Coughlan, J.M. and Yuille, A.L. "The Phase Space of Minimax Entropy Learning". In preparation. 1999.

Field, D. J. "Relations between the statistics of natural images and the response properties of cortical cells". *Journal of the Optical Society* 4,(12), 2379-2394. 1987.

D.C. Knill and W. Richards. (Eds). **Perception as Bayesian Inference**. Cambridge University Press. 1996.

Olshausen, B. A. and Field, D. J. "Emergence of simple-cell receptive field properties by learning a sparse code for natural images". *Nature*. 381, 607-609. 1996.

B.D. Ripley. **Pattern Recognition and Neural Networks**. Cambridge University Press. 1996.

Ruderman, D. and Bialek, W. "Statistics of Natural Images: Scaling in the Woods". *Physical Review Letters*. 73, Number 6,(8 August 1994), 814-817. 1994.

S.C. Zhu, Y. Wu, and D. Mumford. "Minimax Entropy Principle and Its Application to Texture Modeling". Neural Computation. Vol. 9. no. 8. Nov. 1997.

S.C. Zhu and D. Mumford. "Prior Learning and Gibbs Reaction-Diffusion". IEEE Trans. on PAMI vol. 19, no. 11. Nov. 1997.

S-C Zhu, Y-N Wu and D. Mumford. FRAME: Filters, Random field And Maximum Entropy: — Towards a Unified Theory for Texture Modeling. Int'l Journal of Computer Vision 27(2) 1-20, March/April. 1998.